# Signature Verification using a "Siamese" Time Delay Neural Network

Jane Bromley, Isabelle Guyon, Yann LeCun,
Eduard Säckinger and Roopak Shah
AT&T Bell Laboratories
Holmdel, NJ 07733
jbromley@big.att.com

## Abstract

This paper describes an algorithm for verification of signatures written on a pen-input tablet. The algorithm is based on a novel, artificial neural network, called a "Siamese" neural network. This network consists of two identical sub-networks joined at their outputs. During training the two sub-networks extract features from two signatures, while the joining neuron measures the distance between the two feature vectors. Verification consists of comparing an extracted feature vector with a stored feature vector for the signer. Signatures closer to this stored representation than a chosen threshold are accepted, all other signatures are rejected as forgeries.

## 1  INTRODUCTION

The aim of the project was to make a signature verification system based on the NCR 5990 Signature Capture Device (a pen-input tablet) and to use 80 bytes or less for signature feature storage in order that the features can be stored on the magnetic strip of a credit-card.

Verification using a digitizer such as the 5990, which generates spatial coordinates as a function of time, is known as dynamic verification. Much research has been carried out on signature verification. Function-based methods, which fit a function to the pen trajectory, have been found to lead to higher performance while parameter-based methods, which extract some number of parameters from a signa-

ture, make a lower requirement on memory space for signature storage (see Lorette and Plamondon (1990) for comments). We chose to use the complete time extent of the signature, with the preprocessing described below, as input to a neural network, and to allow the network to compress the information. We believe that it is more robust to provide the network with low level features and to allow it to learn higher order features during the training process, rather than making heuristic decisions e.g. such as segmentation into balistic strokes. We have had success with this method previously (Guyon *et al.*, 1990) as have other authors (Yoshimura and Yoshimura, 1992).

## 2   DATA COLLECTION

All signature data was collected using 5990 Signature Capture Devices. They consist of an LCD overlayed with a transparent digitizer. As a guide for signing, a 1 inch by 3 inches box was displayed on the LCD. However all data captured both inside and outside this box, from first pen down to last pen up, was returned by the device. The 5990 provides the trajectory of the signature in Cartesian coordinates as a function of time. Both the trajectory of the pen on the pad and of the pen above the pad (within a certain proximity of the pad) are recorded. It also uses a pen pressure measurement to report whether the pen is touching the writing screen or is in the air.   Forgers usually copy the shape of a signature. Using such a tablet for signature entry means that a forger must copy both dynamic information and the trajectory of the pen in the air. Neither of these are easily available to a forger and it is hoped that capturing such information from signatures will make the task of a forger much harder. Strangio (1976), Herbst and Liu (1977b) have reported that pen up trajectory is hard to imitate, but also less repeatable for the signer. The spatial resolution of signatures from the 5990 is about 300 dots per inch, the time resolution 200 samples per second and the pad's surface is 5.5 inches by 3.5 inches. Performance was also measured using the same data treated to have a lower resolution of 100 dots per inch. This had essentially no effect on the results.

Data was collected in a university and at Bell Laboratories and NCR cafeterias. Signature donors were asked to sign their signature as consistently as possible or to make forgeries.  When producing forgeries, the signer was shown an example of the genuine signature on a computer screen.  The amount of effort made in producing forgeries varied. Some people practiced or signed the signature of people they knew, others made little effort. Hence, forgeries varied from undetectable to obviously different. Skilled forgeries are the most difficult to detect, but in real life a range of forgeries occur from skilled ones to the signatures of the forger themselves.

Except at Bell Labs., the data collection was not closely monitored so it was no surprise when the data was found to be quite noisy. It was cleaned up according to the following rules:

- Genuine signatures must have between 80% and 120% of the strokes of the first signature signed and, if readable, be of the same name as that typed into the data collection system. (The majority of the signatures were donated by residents of North America, and, typical for such signatures, were readable.) The aim of this was to remove signatures for which only

some part of the signature was present or where people had signed another name e.g. Mickey Mouse.

- Forgeries must be an attempt to copy the genuine signature. The aim of this was to remove examples where people had signed completely different names. They must also have 80% to 120% of the strokes of the signature.

- A person must have signed at least 6 genuine signatures or forgeries.

In total, 219 people signed between 10 and 20 signatures each, 145 signed genuines, 74 signed forgeries.

## 3   PREPROCESSING

A signature from the 5990 is typically 800 sets of $x, y$ and pen up-down points. $x(t)$ and $y(t)$ were originally in absolute position coordinates. By calculating the linear estimates for the $x$ and $y$ trajectories as a function of time and subtracting this from the original $x$ and $y$ values, they were converted to a form which is invariant to the position and slope of the signature. Then, dividing by the $y$ standard deviation provided some size normalization (a person may sign their signature in a variety of sizes, this method would normalize them). The next preprocessing step was to resample, using linear interpolation, all signatures to be the same length of 200 points as the neural network requires a fixed input size. Next, further features were computed for input to the network and all input values were scaled so that the majority fell between +1 and −1. Ten different features could be calculated, but a subset of eight were used in different experiments:

**feature 1** pen up $= -1$ ; pen down $= +1$, (pud)

**feature 2** x position, as a difference from the linear estimate for $x(t)$, normalized using the standard deviation of $y$, (x)

**feature 3** y position, as a difference from the linear estimate for $y(t)$, normalized using the standard deviation of $y$, (y)

**feature 4** speed at each point, (spd)

**feature 5** centripetal acceleration, (acc-c)

**feature 6** tangential acceleration, (acc-t)

**feature 7** the direction cosine of the tangent to the trajectory at each point, ($\cos\theta$)

**feature 8** the direction sine of the tangent to the trajectory at each point, ($\sin\theta$)

**feature 9** cosine of the local curvature of the trajectory at each point, ($\cos\phi$)

**feature 10** sine of the local curvature of the trajectory at each point, ($\sin\phi$)

In contrast to the features chosen for character recognition with a neural network (Guyon *et al.*, 1990), where we wanted to eliminate writer specific information, the features such as speed and acceleration were chosen to carry information that aids the discrimination between genuine signatures and forgeries. At the same time we still needed to have some information about shape to prevent a forger from breaking the system by just imitating the rhythm of a signature, so positional, directional amd curvature features were also used. The resampling of the signatures was such as to preserve the regular spacing in time between points. This method penalizes forgers who do not write at the correct speed.

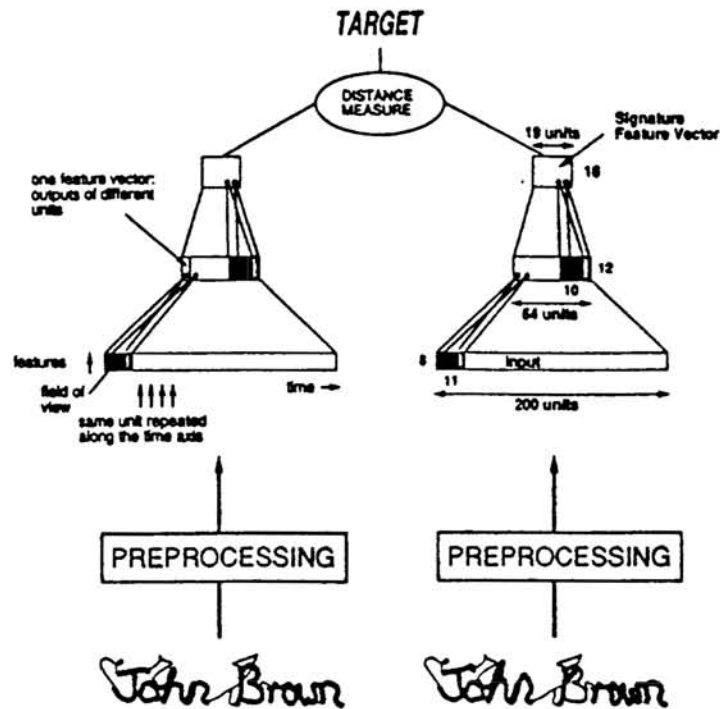

Figure 1: Architecture 1 consists of two identical time delay neural networks. Each network has an input of 8 by 200 units, first layer of 12 by 64 units with receptive fields for each unit being 8 by 11 and a second layer of 16 by 19 units with receptive fields 12 by 10.

## 4    NETWORK ARCHITECTURE AND TRAINING

The Siamese network has two input fields to compare two patterns and one output whose state value corresponds to the similarity between the two patterns. Two separate sub-networks based on Time Delay Neural Networks (Lang and Hinton, 1988, Guyon *et al.* 1990) act on each input pattern to extract features, then the cosine of the angle between two feature vectors is calculated and this represents the distance value. Results for two different subnetworks are reported here.

Architecture 1 is shown in Fig 1. Architecture 2 differs in the number and size of layers. The input is 8 by 200 units, the first convolutional layer is 6 by 192 units with each unit's receptive field covering 8 by 9 units of the input. The first averaging layer is 6 by 64 units, the second convolution layer is 4 by 57 with 6 by 8 receptive fields and the second averaging layer is 4 by 19. To achieve compression in the time dimension, architecture 1 uses a sub-sampling step of 3, while architecture 2 uses averaging. A similar Siamese architecture was independently proposed for fingerprint identification by Baldi and Chauvin (1992).

Training was carried out using a modified version of backpropagation (LeCun, 1989). All weights could be learnt, but the two sub-networks were constrained to have identical weights. The desired output for a pair of genuine signatures was for a small angle (we used cosine=1.0) between the two feature vectors and a large angle

Table 1: Summary of the Training.
Note: GA is the percentage of genuine signature pairs with output greater than 0, FR the percentage of genuine:forgery signature pairs for which the output was less than 0. The aim of removing all pen up points for Network 2 was to investigate whether the pen up trajectories were too variable to be helpful in verification. For Network 4 the training simulation crashed after the 42nd iteration and was not restarted. Performance may have improved if training had continued past this point.

| Network | Input Features | Best Performance on: | |
|---------|----------------|----------------------|---|
| | | Training Set | Validation Set |
| 1, arch 1 | pud acc-c acc-t spd $\cos\theta$ $\sin\theta$ $\cos\phi$ $\sin\phi$ | GA 97.0% FR 65.3%, 26 passes through set | GA 90.3% FR 74.8%, 6 passes through set |
| 2, arch 1 | same as 1, but pen up trajectory removed | GA 97.8% FR 60.0%, 11 passes through set | GA 93.2% FR 75.2%, 2 passes through set |
| 3, arch 1 | x y pud spd $\cos\theta$ $\sin\theta$ $\cos\phi$ $\sin\phi$ | GA 99.8% FR 88.8%, 100 passes through set | GA 91.7% FR 74.2%, 32 passes through set |
| 4, arch 1 | same as network 3, but a larger training set | GA 98.2% FR 81.7%, 42 passes through set | GA 99.4% FR 80.5%, 42 passes through set |
| 5, arch 2 | same as 4, except architecture 2 was used | GA 98.6% FR 81.5%, 69 passes through set | GA 99.6% FR 80.1%, 44 passes through set |

(we used cosine= −0.9 and −1.0) if one of the signatures was a forgery. The training set consisted of 982 genuine signatures from 108 signers and 402 forgeries of about 40 of these signers. We used up to 7,701 signature pairs; 50% genuine:genuine pairs, 40% genuine:forgery pairs and 10% genuine:zero-effort pairs. [1] The validation set consisted of 960 signature pairs in the same proportions as the training set. The network used for verification was that with the lowest error rate on the validation set.

See Table 1 for a summary of the experiments. Training took a few days on a SPARC 1+.

## 5  TESTING

When used for verification, only one sub-network is evaluated. The output of this is the feature vector for the signature. The feature vectors for the last six signatures signed by each person were used to make a multivariate normal density model of the person's signature (see pp. 22–27 of *Pattern Classification and Scene Analysis* by Duda and Hart for a fuller description of this). For simplicity, we assume that the features are statistically independent, and that each feature has the same variance. Verification consists of comparing a feature vector with the model of the signature. The probability density that a test signature is genuine, $\rho$-yes, is found by evaluating

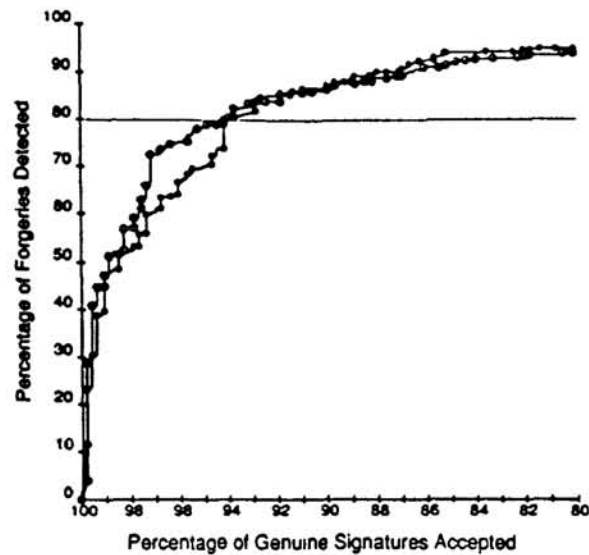

Figure 2: Results for Networks 4 (open circles) and 5 (closed circles). The training of Network 4 was essentially the same as for Network 3 except that more data was used in training and it had been cleaned of noise. They were both based on architecture 1. Network 5 was based on architecture 2. The signature feature vector from this architecture is just 4 by 19 in size.

the normal density function. The probability density that a test signature is a forgery, $\rho$-no, is assumed, for simplicity, to be a constant value over the range of interest. An estimate for this value was found by averaging the $\rho$-yes values for all forgeries. Then the probability that a test signature is genuine is $\rho$-yes/($\rho$-yes + $\rho$-no). Signatures closer than a chosen threshold to this stored representation are accepted, all other signatures are rejected as forgeries.

Networks 1, 2 and 3, all based on architecture 1, were tested using a set of 63 genuine signatures and 63 forgeries for 18 different people. There were about 4 genuine test signatures for each of the 18 people, and 10 forgeries for 6 of these people. Networks 1 and 2 had identical training except Network 2 was trained without pen up points. Network 1 gave the better results. However, with such a small test set, this difference may be hardly significant.

The training of Network 3 was identical to that of Network 1, except that x and y were used as input features, rather than acc-c and acc-t. It had somewhat improved performance. No study was made to find out whether the performance improvement came from using x and y or from leaving out acc-c and acc-t. Plamondon and Parizeau (1988) have shown that acceleration is not as reliable as other functions.

Figure 2 shows the results for Networks 4 and 5. They were tested using a set of 532 genuine signatures and 424 forgeries for 43 different people. There were about 12 genuine test signatures for each person, and 30 forgeries for 14 of the people. This graph compares the performance of the two different architectures.

It takes 2 to 3 minutes on a Sun SPARC2 workstation to preprocess 6 signatures,

collect the 6 outputs from the sub-network and build the normal density model.

# 6   RESULTS

Best performance was obtained with Network 4. With the threshold set to detect 80% of forgeries, 95.5% of genuine signatures were detected (24 signatures rejected). Performance could be improved to 97.0% genuine signatures detected (13 rejected) by removing all first and second signature from the test set [2]. For 9 of the remaining 13 rejected signatures pen up trajectories differed from the person's typical signature. This agrees with other reports (Strangio, 1976 Herbst and Liu, 1977b) that pen up trajectory is hard to imitate, but also a less repeatable signature feature. However, removing pen up trajectories from training and test sets did not lead to any improvement (Networks 1 and 2 had similar performance), leading us to believe that pen up trajectories are useful in some cases. Using an elastic matching method for measuring distance may help. Another cause of error came from a few people who seemed unable to sign consistently and would miss out letters or add new strokes to their signature.

The requirement to represent a model of a signature in 80 bytes means that the signature feature vector must be encodable in 80 bytes. Architecture 2 was specifically designed with this requirement in mind. Its signature feature vector has 76 dimensions. When testing Network 5, which was based on this architecture, 50% of the outputs were found (surprisingly) to be redundant and the signature could be represented by a 38 dimensional vector with no loss of performance. One explanation for this redundancy is that, by reducing the dimension of the output (by not using some outputs), it is easier for the neural network to satisfy the constraint that genuine and forgery vectors have a cosine distance of $-1$ (equivalent to the outputs from two such signatures pointing in opposite directions).

These results were gathered on a Sun SPARC2 workstation where the 38 values were each represented with 4 bytes. A test was made representing each value in one byte. This had no detrimental effect on the performance. Using one byte per value allows the signature feature vector to be coded in 38 bytes, which is well within the size constraint. It may be possible to represent a signature feature vector with even less resolution, but this was not investigated. For a model to be updatable (a requirement of this project), the total of all the squares for each component of the signature feature vectors must also be available. This is another 38 dimensional vector. From these two vectors the variance can be calculated and a test signature verified. These two vectors can be stored in 80 bytes.

# 7   CONCLUSIONS

This paper describes an algorithm for signature verification. A model of a person's signature can easily fit in 80 bytes and the model can be updated and become more accurate with each successful use of the credit card (surely an incentive for people to use their credit card as frequently as possible). Other beneficial aspects of this verification algorithm are that it is more resistant to forgeries for people who sign

consistently, the algorithm is independent of the general direction of signing and is insensitive to changes in size and slope.

As a result of this project, a demonstration system incorporating the neural network signature verification algorithm was developed. It has been used in demonstrations at Bell Laboratories where it worked equally well for American, European and Chinese signatures. This has been shown to commercial customers. We hope that a field trial can be run in order to test this technology in the real world.

## Acknowledgements

All the neural network training and testing was carried out using SN2.6, a neural network simulator package developed by Neuristique. We would like to thank Bernhard Boser, John Denker, Donnie Henderson, Vic Nalwa and the members of the Interactive Systems department at AT&T Bell Laboratories, and Cliff Moore at NCR Corporation, for their help and encouragement. Finally, we thank all the people who took time to donate signatures for this project.

## Footnotes

[1] zero-effort forgeries, also known as random forgeries, are those for which the forger makes no effort to copy the genuine signature, we used genuine signatures from other signers to simulate such forgeries.

[2]people commented that they needed to sign a few time to get accustomed to the pad

## References

P. Baldi and Y. Chauvin, "Neural Networks for Fingerprint Recognition", *Neural Computation*, **5** (1993).

R. Duda and P. Hart, *Pattern Classification and Scene Analysis*, John Wiley and Sons, Inc., 1973.

I. Guyon, P. Albrecht, Y. LeCun, J. S. Denker and W. Hubbard, "A Time Delay Neural Network Character Recognizer for a Touch Terminal", *Pattern Recognition*, (1990).

N. M. Herbst and C. N. Liu, "Automatic signature verification based on accelerometry", *IBM J. Res. Develop.*, **21** (1977)245–253.

K. J. Lang and G. E. Hinton, "A Time Delay Neural Network Architecture for Speech Recognition", Technical Report CMU-cs-88-152, Carnegie-Mellon University, Pittsburgh, PA, 1988.

Y. LeCun, "Generalization and Network Design Strategies", Technical Report CRG-TR-89-4 University of Toronto Connectionist Research Group, Canada, 1989.

G. Lorette and R. Plamondon, "Dynamic approaches to handwritten signature verification", in *Computer processing of handwriting*, Eds. R. Plamondon and C. G. Leedham, World Scientific, 1990.

R. Plamondon and M. Parizeau, "Signature verification from position, velocity and acceleration signals: a comparative study", in *Proc. 9th Int. Con. on Pattern Recognition*, Rome, Italy, 1988, pp 260–265.

C. E. Strangio, "Numerical comparison of similarly structured data perturbed by random variations, as found in handwritten signatures", Technical Report, Dept. of Elect. Eng., 1976.

I. Yoshimura and M. Yoshimura, "On-line signature verification incorporating the direction of pen movement – an experimental examination of the effectiveness", in *From pixels to features III: frontiers in Handwriting recognition*, Eds. S. Impedova and J. C. Simon, Elsevier, 1992.
